# Kernel Change-point Analysis

**Zaïd Harchaoui**
LTCI, TELECOM ParisTech and CNRS
46, rue Barrault, 75634 Paris cedex 13, France
zaid.harchaoui@enst.fr

**Francis Bach**
Willow Project, INRIA-ENS
45, rue d'Ulm, 75230 Paris, France
francis.bach@mines.org

**Éric Moulines**
LTCI, TELECOM ParisTech and CNRS
46, rue Barrault, 75634 Paris cedex 13, France
eric.moulines@enst.fr

## Abstract

We introduce a kernel-based method for change-point analysis within a sequence of temporal observations. Change-point analysis of an unlabelled sample of observations consists in, first, testing whether a change in the distribution occurs within the sample, and second, if a change occurs, estimating the change-point instant after which the distribution of the observations switches from one distribution to another different distribution. We propose a test statistic based upon the maximum kernel Fisher discriminant ratio as a measure of homogeneity between segments. We derive its limiting distribution under the null hypothesis (no change occurs), and establish the consistency under the alternative hypothesis (a change occurs). This allows to build a statistical hypothesis testing procedure for testing the presence of a change-point, with a prescribed false-alarm probability and detection probability tending to one in the large-sample setting. If a change actually occurs, the test statistic also yields an estimator of the change-point location. Promising experimental results in temporal segmentation of mental tasks from BCI data and pop song indexation are presented.

## 1 Introduction

The need to partition a sequence of observations into several homogeneous segments arises in many applications, ranging from speaker segmentation to pop song indexation. So far, such tasks were most often dealt with using probabilistic sequence models, such as hidden Markov models [1], or their discriminative counterparts such as conditional random fields [2]. These probabilistic models require a sound knowledge of the transition structure between the segments and demand careful training beforehand to yield competitive performance; when data are acquired online, inference in such models is also not straightforward (see, e.g., [3, Chap. 8]). Such models essentially perform multiple change-point *estimation*, while one is often also interested in meaningful quantitative measures for the *detection* of a change-point within a sample.

When a parametric model is available to model the distributions before and after the change, a comprehensive literature for change-point analysis has been developed, which provides optimal criteria from the maximum likelihood framework, as described in [4]. Nonparametric procedures were also proposed, as reviewed in [5], but were limited to univariate data and simple settings. Online counterparts have also been proposed and mostly built upon the cumulative sum scheme (see [6] for extensive references). However, so far, even extensions to the case where the distribution before the change is known, and the distribution after the change is not known, remains an open problem [7]. This brings to light the need to develop statistically grounded change-point analysis algorithms, working on multivariate, high-dimensional, and also structured data.

We propose here a regularized kernel-based test statistic, which allows to simultaneously provide quantitative answers to both questions: 1) is there a change-point within the sample? 2) if there is one, then where is it? We prove that our test statistic for change-point analysis has a false-alarm probability tending to $\alpha$ and a detection probability tending to one as the number of observations tends to infinity. Moreover, the test statistic directly provides an accurate estimate of the change-point instant. Our method readily extends to multiple change-point settings, by performing a sequence of change-point analysis in *sliding windows* running along the signal. Usually, physical considerations allow to set the window-length to a sufficiently small length for being guaranteed that *at most one change-point* occurs within each window, and sufficiently large length for our decision rule to be statistically significant (typically $n > 50$).

In Section 2, we set up the framework of change-point analysis, and in Section 3, we describe how to devise a regularized kernel-based approach to the change-point problem. Then, in Section 4 and in Section 5, we respectively derive the limiting distribution of our test statistic under the null hypothesis $\mathbf{H}_0$ : "no change occurs", and establish the consistency in power under the alternative $\mathbf{H}_A$ : "a change occurs". These theoretical results allow to build a test statistic which has provably a false-alarm probability tending to a prescribed level $\alpha$, and a detection probability tending to one, as the number of observations tends to infinity. Finally, in Section 7, we display the performance of our algorithm for respectively, segmentation into mental tasks from BCI data and temporal segmentation of pop songs.

## 2   Change-point analysis

In this section, we outline the change-point problem, and describe formally a strategy for building change-point analysis test statistics.

**Change-point problem**     Let $X_1, \ldots, X_n$ be a time series of *independent* random variables. The change-point analysis of the sample $\{X_1, \ldots, X_n\}$ consists in the following two steps.

1) Decide between

$$\mathbf{H}_0 : \quad \mathbb{P}_{X_1} = \cdots = \mathbb{P}_{X_k} = \cdots = \mathbb{P}_{X_n}$$
$$\mathbf{H}_A : \quad \text{there exists } 1 < k^\star < n \text{ such that} \tag{1}$$
$$\mathbb{P}_{X_1} = \cdots = \mathbb{P}_{X_{k^\star}} \neq \mathbb{P}_{X_{k^\star+1}} = \cdots = \mathbb{P}_{X_n} \ .$$

2) Estimate $k^\star$ from the sample $\{X_1, \ldots, X_n\}$ if $\mathbf{H}_A$ is true .

While sharing many similarities with usual machine learning problems, the change-point problem is different.

**Statistical hypothesis testing**   An important aspect of the above formulation of the change-point problem is its natural embedding in a statistical hypothesis testing framework. Let us recall briefly the main concepts in statistical hypothesis testing, in order to rephrase them within the change-point problem framework (see, e.g., [8]). The goal is to build a decision rule to answer question 1) in the change-point problem stated above. Set a *false-alarm probability* $\alpha$ with $0 < \alpha < 1$ (also called level or Type I error), whose purpose is to theoretically guarantee that $\mathbb{P}(\text{decide } \mathbf{H}_A, \text{ when } \mathbf{H}_0 \text{ is true})$ is close to $\alpha$. Now, if there actually is a change-point within the sample, one would like not to miss it, that is that the *detection probability* $\pi = \mathbb{P}(\text{decide } \mathbf{H}_A, \text{ when } \mathbf{H}_A \text{ is true})$—also called power and equal to one minus the Type II error—should be close to one. The purpose of Sections 4-5 is to give theoretical guarantees to those practical requirements in the large-sample setting, that is when the number of observations $n$ tends to infinity.

**Running maximum partition strategy**   An efficient strategy for building change-point analysis procedures is to select the partition of the sample which yields a maximum heterogeneity between the two segments: given a sample $\{X_1, \ldots, X_n\}$ and a candidate change point $k$ with $1 < k < n$, assume we may compute a measure of heterogeneity $\Delta_{n,k}$ between the segments $\{X_1, \ldots, X_k\}$ on the one hand, and $\{X_{k+1}, \ldots, X_n\}$ on the other hand. Then, the "running maximum partition strategy" consists in using $\max_{1<k<n} \Delta_{n,k}$ as a building block for change-point analysis (cf. Figure 1). Not only $\max_{1<k<n} \Delta_{n,k}$ may be used to test for the *presence* of a change-point and assess/discard

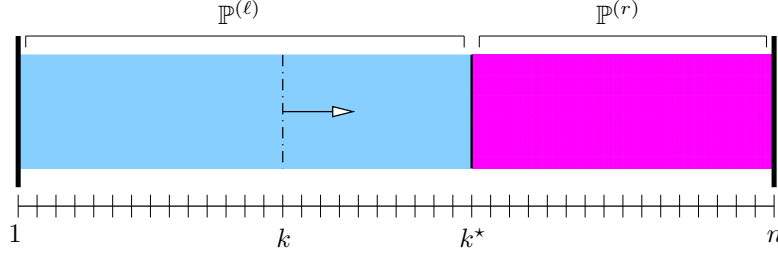

Figure 1: The running maximum strategy for change-point analysis. The test statistic for change-point analysis runs a candidate change-point $k$ with $1 < k < n$ along the sequence of observations, hoping to catch the true change-point $k^\star$.

the overall homogeneity of the sample; besides, $\hat{k} = \operatorname{argmax}_{1<k<n}\Delta_{n,k}$ provides a sensible estimator of the true change-point instant $k^\star$ [5].

## 3 Kernel Change-point Analysis

In this section, we describe how the kernel Fisher discriminant ratio, which has proven relevant for measuring the homogeneity of two samples in [9], may be embedded into the *running maximum partition strategy* to provide a powerful test statistic, coined KCpA for **K**ernel **C**hange-**p**oint **A**nalysis, for addressing the change-point problem. This is described in the operator-theoretic framework, developed for the statistical analysis of kernel-based learning and testing algorithms in [10, 11].

**Reproducing kernel Hilbert space** Let $(\mathcal{X}, d)$ be a separable measurable metric space. Let $X$ be an $\mathcal{X}$-valued random variable, with probability measure $\mathbb{P}$; the expectation with respect to $\mathbb{P}$ is denoted by $\mathbb{E}[\cdot]$ and the covariance by $\operatorname{Cov}(\cdot, \cdot)$. Consider a reproducing kernel Hilbert space (RKHS) $(\mathcal{H}, \langle\cdot, \cdot\rangle_{\mathcal{H}})$ of functions from $\mathcal{X}$ to $\mathbb{R}$. To each point $x \in \mathcal{X}$, there corresponds an element $\Phi(x) \in \mathcal{H}$ such that $\langle\Phi(x), f\rangle_{\mathcal{H}} = f(x)$ for all $f \in \mathcal{H}$, and $\langle\Phi(x), \Phi(y)\rangle_{\mathcal{H}} = k(x, y)$, where $k : \mathcal{X} \times \mathcal{X} \to \mathbb{R}$ is a positive definite kernel [12]. In the following, we exclusively work with the Aronszajn-map, that is, we take $\Phi(x) = k(x, \cdot)$ for all $x \in \mathcal{X}$. It is assumed from now on that $\mathcal{H}$ is a separable Hilbert space. Note that this is always the case if $\mathcal{X}$ is a separable metric space and if the kernel is continuous [13]. We make the following two assumptions on the kernel (which are satisfied in particular for the Gaussian kernel; see [14]): **(A1)** the kernel $k$ is bounded, that is $\sup_{(x,y)\in\mathcal{X}\times\mathcal{X}} k(x, y) < \infty$, **(A2)** for all probability distributions $\mathbb{P}$ on $\mathcal{X}$, the RKHS associated with $k(\cdot, \cdot)$ is dense in $L^2(\mathbb{P})$.

**Kernel Fisher Discriminant Ratio** Consider a sequence of independent observations $X_1, \ldots, X_n \in \mathcal{X}$. For any $[i, j] \subset \{2, \ldots, n-1\}$, define the corresponding empirical mean elements and covariance operators as follows

$$\hat{\mu}_{i:j} := \frac{1}{j-i+1}\sum_{\ell=i}^{j} k(X_\ell, \cdot)\,, \quad \hat{\Sigma}_{i:j} := \frac{1}{j-i+1}\sum_{\ell=i}^{j}\{k(X_\ell, \cdot) - \hat{\mu}_{i:j}\} \otimes \{k(X_\ell, \cdot) - \hat{\mu}_{i:j}\}\,.$$

These quantities have obvious population counterparts, the population mean element and the population covariance operator, defined for any probability measure $\mathbb{P}$ as $\langle\mu_{\mathbb{P}}, f\rangle_{\mathcal{H}} := \mathbb{E}[f(X)]$ for all $f \in \mathcal{H}$, and $\langle f, \Sigma_{\mathbb{P}}g\rangle_{\mathcal{H}} := \operatorname{Cov}_{\mathbb{P}}[f(X), g(X)]$ for $f, g \in \mathcal{H}$. For all $k \in \{2, \ldots, n-1\}$ the (maximum) *kernel Fisher discriminant ratio*, which we abbreviate as KFDR is defined as

$$\mathsf{KFDR}_{n,k;\gamma}(X_1, \ldots, X_n) := \frac{k(n-k)}{n} \left\| \left(\frac{k}{n}\hat{\Sigma}_{1:k} + \frac{n-k}{n}\hat{\Sigma}_{k+1:n} + \gamma\mathrm{I}\right)^{-1/2}(\hat{\mu}_{k+1:n} - \hat{\mu}_{1:k}) \right\|_{\mathcal{H}}^2.$$

Note that, if we merge two labelled samples $\{X_1, \ldots, X_{n_1}\}$ and $\{X'_1, \ldots, X'_{n_2}\}$ into a single sample as $\{X_1, \ldots, X_{n_1}, X'_1, \ldots, X'_{n_2}\}$, then with $\mathsf{KFDR}_{n_1+n_2,n_1+1;\gamma}(X_1, \ldots, X_{n_1}, X'_1, \ldots, X'_{n_2})$ we recover the test statistic considered in [9] for testing the homogeneity of two samples $\{X_1, \ldots, X_{n_1}\}$ and $\{X'_1, \ldots, X'_{n_2}\}$.

Following [9], we make the following assumptions on all the covariance operators $\Sigma$ considered in this paper: (**B1**) the eigenvalues $\{\lambda_p(\Sigma)\}_{p\geq 1}$ satisfy $\sum_{p=1}^{\infty} \lambda_p^{1/2}(\Sigma) < \infty$, (**B2**) there are infinitely many strictly positive eigenvalues $\{\lambda_p(\Sigma)\}_{p\geq 1}$ of $\Sigma$.

**Kernel change-point analysis**     Now, we may apply the strategy described before (cf. Figure 1) to obtain the main building block of our test statistic for change-point analysis. Indeed, we define our test statistic $T_{n,k;\gamma}$ as

$$ T_{n;\gamma}(k) := \max_{a_n < k < b_n} \frac{\mathsf{KFDR}_{n,k;\gamma} - d_{1,n,k;\gamma}(\hat{\Sigma}_{n,k}^W)}{\sqrt{2}\, d_{2,n,k;\gamma}(\hat{\Sigma}_{n,k}^W)} \,, $$

where $n\hat{\Sigma}_{n,k}^W := k\hat{\Sigma}_{1:k} + (n-k)\hat{\Sigma}_{k+1:n}$. The quantities $d_{1,n,k;\gamma}(\hat{\Sigma}_{n,k}^W)$ and $d_{2,n,k;\gamma}(\hat{\Sigma}_{n,k}^W)$, defined respectively as

$$ d_{1,n,k;\gamma}(\hat{\Sigma}_{n,k}^W) := \mathrm{Tr}\{(\hat{\Sigma}_{n,k}^W + \gamma\mathrm{I})^{-1}\hat{\Sigma}_{n,k}^W\} \,, \quad d_{2,n,k;\gamma}(\hat{\Sigma}_{n,k}^W) := \mathrm{Tr}\{(\hat{\Sigma}_{n,k}^W + \gamma\mathrm{I})^{-2}(\hat{\Sigma}_{n,k}^W)^2\} \,, $$

act as normalizing constants for $T_{n;\gamma}(k)$ to have zero-mean and unit-variance as $n$ tends to infinity, a standard statistical transformation known as *studentization*. The maximum is searched within the interval $[a_n, b_n]$ with $a_n > 1$ and $b_n < n$, which is restriction of $]1, n[$, in order to prevent the test statistic from uncontrolled behaviour in the neighborhood of the interval boundaries, which is standard practice in this setting [15].

**Remark**     Note that, if the input space is Euclidean, for instance $\mathcal{X} = \mathbb{R}^d$, and if the kernel is linear $k(x,y) = x^T y$, then $T_{n;\gamma}(k)$ may be interpreted as a regularized version of the classical maximum-likelihood multivariate test statistic used to test change in mean with unequal covariances, under the assumption of normal observations, described in [4, Chap. 3]. Yet, as the next section shall show, our test statistic is truly nonparametric, and its large-sample properties *do not require* any "gaussian in the feature space"-type of assumption. Moreover, in practice it may be computed thanks to the kernel trick, adapted to the kernel Fisher discriminant analysis and outlined in [16, Chapter 6].

**False-alarm and detection probability**     In order to build a principled testing procedure, a proper theoretical analysis from a statistical point of view is necessary. First, as the next section shows, for a prescribed $\alpha$, we may build a procedure which has, as $n$ tends to infinity, the false-alarm probability $\alpha$ under the null hypothesis $\mathbf{H}_0$, that is when the sample is completely homogeneous and contains no-change-point. Besides, when the sample actually contains at most one change-point, we prove that our test statistic is able to catch it with probability one as $n$ tends to infinity.

**Large-sample setting**     For the sake of generality, we describe here the large-sample setting for the change-point problem under the alternative hypothesis $\mathbf{H}_A$. Essentially, it corresponds to normalizing the signal sampling interval to $[0, 1]$ and letting the resolution increase as we observe more data points [4].

Assume there is $0 < k^\star < n$ such that $\mathbb{P}_{X_1} = \cdots = \mathbb{P}_{X_{k^\star}} \neq \mathbb{P}_{X_{k^\star+1}} = \cdots = \mathbb{P}_{X_n}$. Define $\tau^\star := k^\star/n$ such that $\tau^\star \in ]0, 1[$, and define $\mathbb{P}^{(\ell)}$ the probability distribution prevailing within the left segment of length $\tau^\star$, and $\mathbb{P}^{(r)}$ the probability distribution prevailing within the right segment of length $1 - \tau^\star$. Then, we want to study what happens if we have $\lfloor n\tau^\star \rfloor$ observations from $\mathbb{P}^{(\ell)}$ (before change) and $\lfloor n(1 - \tau^\star) \rfloor$ observations from $\mathbb{P}^{(r)}$ (after change) where $n$ is large and $\tau^\star$ is kept fixed.

## 4   Limiting distribution under the null hypothesis

Throughout this section, we work under the null hypothesis $\mathbf{H}_0$ that is $\mathbb{P}_{X_1} = \cdots = \mathbb{P}_{X_k} = \cdots = \mathbb{P}_{X_n}$ for all $2 \leq k \leq n$. The first result gives the limiting distribution of $T_{n;\gamma}(k)$ as the number of observations $n$ tends to infinity.

Before stating the theoretical results, let us describe informally the crux of our approach. We may prove, under $\mathbf{H}_0$, using operator-theoretic pertubation results similar to [9], that it is sufficient to study the large-sample behaviour of $\tilde{T}_{n;\gamma}(k) := \max_{a_n < k < b_n}(\sqrt{2}\, d_{2;\gamma}(\Sigma))^{-1}Q_{n,\infty;\gamma}(k)$ where

$$ Q_{n,\infty;\gamma}(k) := \frac{k(n-k)}{n} \left\| (\Sigma + \gamma\mathrm{I})^{-1/2}(\hat{\mu}_{k+1:n} - \hat{\mu}_{1:k}) \right\|_{\mathcal{H}}^2 - d_{1;\gamma}(\Sigma) \,, \quad 1 < k < n \,, \quad (2) $$

and $d_{1;\gamma}(\Sigma)$ and $d_{2;\gamma}(\Sigma)$ are respectively the population recentering and rescaling quantities with $\Sigma = \Sigma_{1:n} = \Sigma_{1:n}^W$ the within-class covariance operator. Note that the only remaining stochastic term in (2) is $\hat{\mu}_{k+1:n} - \hat{\mu}_{1:k}$. Let us expand (2) onto the eigenbasis $\{\lambda_p, e_p\}_{p \geq 1}$ of the covariance operator $\Sigma$, as follows:

$$Q_{n,\infty;\gamma}(k) = \sum_{p=1}^{\infty} (\lambda_p + \gamma)^{-1} \left\{ \frac{k(n-k)}{n} \langle \mu_{k+1:n} - \mu_{1:k}, e_p \rangle^2 - \lambda_p \right\} , \quad 1 < k < n . \quad (3)$$

Then, defining $S_{1:k,p} := n^{-1/2} \sum_{i=1}^{k} \lambda_p^{-1/2} (e_p(X_i) - \mathbb{E}[e_p(X_1)])$, we may rewrite $Q_{n,\infty;\gamma}(k)$ as an infinite-dimensional quadratic form in the tied-down partial sums $S_{1:k,p} - \frac{k}{n} S_{1:n,p}$, which yields

$$Q_{n,\infty;\gamma}(k) = \sum_{p=1}^{\infty} (\lambda_p + \gamma)^{-1} \lambda_p \left\{ \frac{n^2}{k(n-k)} \left( S_{1:k,p} - \frac{k}{n} S_{1:n,p} \right)^2 - 1 \right\} , \quad 1 < k < n . \quad (4)$$

The idea is to view $\{Q_{n,\infty;\gamma}(k)\}_{1<k<n}$ as a stochastic process, that is a random function $[k \mapsto Q_{n,\infty;\gamma}(k,\omega)]$ for any $\omega \in \Omega$, where $(\Omega, \mathcal{F}, \mathbb{P})$ is a probability space. Then, invoking the so-called *invariance principle in distribution* [17], we realize that the random sum $S_{1:\lfloor nt \rfloor, p}(\omega)$, which for all $\omega$ linearly interpolates between the values $S_{1:i/n,p}(\omega)$ at points $i/n$ for $i = 1, \ldots, n$, behaves, asymptotically as $n$ tends to infinity, like a Brownian motion (also called **W**iener process) $\{\mathbf{W}_p(t)\}_{0<t<1}$. Hence, along each component $e_p$, we may define a Brownian bridge $\{\mathbf{B}_p(t)\}_{0<t<1}$, that is a tied-down brownian motion $\mathbf{B}_p(t) := \mathbf{W}_p(t) - t\mathbf{W}_p(1)$ which yields continuous approximation *in distribution* of the corresponding $\{S_{1:k,p} - \frac{k}{n} S_{1:n,p}\}_{1<k<n}$. The proof (omitted due to space limitations) consists in deriving a functional (noncentral) limit theorem for $\mathsf{KFDR}_{n,k;\gamma}$, and then applying a continuous mapping argument.

**Proposition 1** *Assume (A1) and (B1), and that $\boldsymbol{H}_0$ holds, that is $\mathbb{P}_{X_i} = \mathbb{P}$ for all $1 \leq i \leq n$. Assume in addition that the regularization parameter $\gamma$ is held fixed as $n$ tends to infinity, and that $a_n/n \to u > 0$ and $b_n/n \to v < 1$ as $n$ tends to infinity. Then,*

$$T_{n;\gamma}(k) \xrightarrow{\mathcal{D}} \sup_{u<t<v} Q_{\infty;\gamma}(t) := \frac{1}{\sqrt{2} d_{2;\gamma}(\Sigma)} \sum_{p=1}^{\infty} \frac{\lambda_p(\Sigma)}{\lambda_p(\Sigma) + \gamma} \left( \frac{\mathbf{B}_p^2(t)}{t(1-t)} - 1 \right) ,$$

*where $\{\lambda_p(\Sigma)\}_{p \geq 1}$ is the sequence of eigenvalues of the overall covariance operator $\Sigma$, while $\{\mathbf{B}_p(t)\}_{p \geq 1}$ is a sequence of independent brownian bridges.*

Define $t_{1-\alpha}$ as the $(1-\alpha)$-quantile of $\sup_{u<t<v} Q_{\infty;\gamma}(t)$. We may compute $t_{1-\alpha}$ either by Monte-Carlo simulations, as described in [18], or by bootstrap resampling under the null hypothesis (see). The next result proves that, using the limiting distribution under the null stated above, we may build a test statistic with prescribed false-alarm probability $\alpha$ for large $n$.

**Corollary 2** *The test $\max_{a_n<k<b_n} T_{n,\gamma}(k) \geq t_{1-\alpha}(\Sigma, \gamma)$ has false-alarm probability $\alpha$, as $n$ tends to infinity.*

Besides, when the sequence of regularization parameters $\{\gamma_n\}_{n \geq 1}$ decreases to zero slowly enough (in particular slower than $n^{-1/2}$), the test statistic $\max_{a_n<k<b_n} T_{n,\gamma_n}(k)$ turns out to be asymptotically *kernel-independent* as $n$ tends to infinity. While the proof hinges upon martingale functional limit theorems [17], still, we may point out that if we replace $\gamma$ by $\gamma_n$ in the limiting null distribution, then $Q_{\infty;\gamma}(\cdot)$ is correctly normalized for all $n \geq 1$ to have zero-mean and variance one.

**Proposition 3** *Assume (A1) and (B1-B2) and that $\boldsymbol{H}_0$ holds, that is $\mathbb{P}_{X_i} = \mathbb{P}$ for all $1 \leq i \leq n$. Assume in addition that the regularization parameters $\{\gamma_n\}_{n \geq 1}$ is such that*

$$\gamma_n + \frac{d_{1,n;\gamma_n}(\Sigma)}{d_{2,n;\gamma_n}(\Sigma)} \gamma_n^{-1} n^{-1/2} \to 0 ,$$

*and that $a_n/n \to u > 0$ and $b_n/n \to v < 1$ as $n$ tends to infinity. Then,*

$$\max_{a_n<k<b_n} T_{n;\gamma_n}(k) \xrightarrow{\mathcal{D}} \sup_{u<t<v} \frac{\mathbf{B}(t)}{\sqrt{t(1-t)}} .$$

**Remark**    A closer look at Proposition 1 brings to light that the reweighting by $t(1-t)$ of the squared brownian bridges on each component is crucial for our test statistic to be immune against imbalance between segment lengths under the alternative $\mathbf{H}_A$, that is when $\tau^\star$ is far from $1/2$. Indeed, swapping out the reweighting by $t(1-t)$, to simply consider the corresponding unweighted test statistic is hazardous, and yields a loss of power for alternatives when $\tau^\star$ is far from $1/2$.

This section allowed us get an $\alpha$-level test statistic for the change-point problem, by looking at the large-sample behaviour of the test statistic under the null hypothesis $\mathbf{H}_0$. The next step is to prove that the test statistic is *consistent in power*, that is the detection probability tends to one as $n$ tends to infinity under the alternative hypothesis $\mathbf{H}_A$.

# 5    Consistency in power

This section shows that, when the alternative hypothesis $\mathbf{H}_A$ holds, our test statistic is able to detect presence of a change with probability one in the large-sample setting. The next proposition is proved within the same framework as the one considered in the previous section, except that now, along each component $e_p$, one has to split the random sum into three parts $[1, k], [k+1, k^\star], [k^\star + 1, n]$, and then the large-sample behaviour of each projected random sum is the one of a two-sided Brownian motion with drifts.

**Proposition 4** *Assume (A1-A2) and (B1-B2), and that $\mathbf{H}_A$ holds, that is there is exists $u < \tau^\star < v$ with $u > 0$ and $v < 1$ such that $\mathbb{P}_{X_{\lfloor n\tau^\star \rfloor}} \neq \mathbb{P}_{X_{\lfloor n\tau^\star \rfloor + 1}}$ for all $1 \leq i \leq n$. Assume in addition that the regularization parameter $\gamma$ is held fixed as $n$ tends to infinity, and that $\lim_{n\to\infty} a_n/n > u$ and $\lim_{n\to\infty} b_n/n < v$. Then, for any $0 < \alpha < 1$, we have*

$$\mathbb{P}_{\mathbf{H}_A}\left(\max_{a_n < k < b_n} T_{n;\gamma}(k) > t_{1-\alpha}\right) \to 1, \quad as\ n \to \infty. \tag{5}$$

# 6    Extensions and related works

**Extensions**    It is worthwhile to note that we may also have built similar procedures from the maximum mean discrepancy (MMD) test statistic proposed by [19]. Note also that, instead of the Tikhonov-type regularization of the covariance operator, other regularization schemes may also be applied, such as the spectral truncation regularization of the covariance operator, equivalent to pre-processing by a centered kernel principal component analysis [20, 21], as used in [22] for instance.

**Related works**    A related problem is the abrupt change detection problem, explored in [23], which is naturally also encompassed by our framework. Here, one is interested in the early detection of a change from a nominal distribution to an erratic distribution. The procedure KCD of [23] consists in running a window-limited detection algorithm, using two one-class support vector machines trained respectively on the left and the right part of the window, and comparing the sets of obtained weights; Their approach differs from our in two points. First, we have the limiting null distribution of KCpA, which allows to compute decision thresholds in a principled way. Second, our test statistic incorporates a reweighting to keep power against alternatives with unbalanced segments.

# 7    Experiments

**Computational considerations**    In all experiments, we set $\gamma = 10^{-5}$ and took the Gaussian kernel with isotropic bandwidth set by the plug-in rule used in density estimation. Second, since from $k$ to $k+1$, the test statistic changes from $\mathsf{KFDR}_{n,k;\gamma}$ to $\mathsf{KFDR}_{n,k+1;\gamma}$, it corresponds to take into account the change from $\{(X_1, Y_1 = -1), \ldots, (X_k, Y_k = -1), (X_{k+1}, Y_{k+1} = +1), \ldots, (X_n, Y_n = +1)\}$ to $\{(X_1, Y_1 = -1), \ldots, (X_k, Y_k = -1), (X_{k+1}, Y_{k+1} = -1), (X_{k+2}, Y_{k+2} = +1) \ldots, (X_n, Y_n = +1)\}$ in the labelling in $\mathsf{KFDR}$ [9, 16]. This motivates an efficient strategy for the computation of the test statistic. We compute the matrix inversion of the regularized kernel gram matrix once for all, at the cost of $O(n^3)$, and then compute all values of the test statistic for all partitions in one matrix multiplication—in $O(n^2)$. As for computing the decision threshold $t_{1-\alpha}$, we used bootstrap resampling calibration with $10,000$ runs. Other Monte-Carlo based calibration procedures are possible, but are left for future research.

|  | Subject 1 | Subject 2 | Subject 3 |
|---|---|---|---|
| KCpA | 79% | 74% | 61% |
| SVM | 76% | 69% | 60% |

Table 1: Average classification accuracy for each subject

**Brain-computer interface data**     Signals acquired during Brain-Computer Interface (BCI) trial experiments naturally exhibit temporal structure. We considered a dataset proposed in BCI competition III[1] acquired during 4 non-feedback sessions on 3 normal subjects, where each subject was asked to perform different tasks, the time where the subject switches from one task to another being random (see also [24]). Mental tasks segmentation is usually tackled with supervised classification algorithms, which require labelled data to be acquired beforehand. Besides, standard supervised classification algorithms are context-sensitive, and sometimes yield poor performance on BCI data. We performed a sequence of change-point analysis on sliding windows overlapping by $20\%$ along the signals. We provide here two ways of measuring the performance of our method. First, in Figure 2 (left), we give in the *empirical ROC-curve* of our test statistic, averaged over all the signals at hand. This shows that our test statistic yield competitive performance for testing the presence of a change-point, when compared with a standard parametric multivariate procedure (param) [4]. Second, in Table 1, we give experimental results in terms of *classification accuracy*, which proves that we can reach comparable/better performance as *supervised* multi-class (one-versus-one) classification algorithms (SVM) with our completely *unsupervised* kernel change-point analysis algorithm. If each segment is considered as a sample of a given class, then the classification accuracy corresponds here to the proportion of correctly assigned points at the end of the segmentation process. This also clearly shows that KCpA algorithm give accurate estimates of the change-points, since the change-point estimation error is directly measured by the classification accuracy.

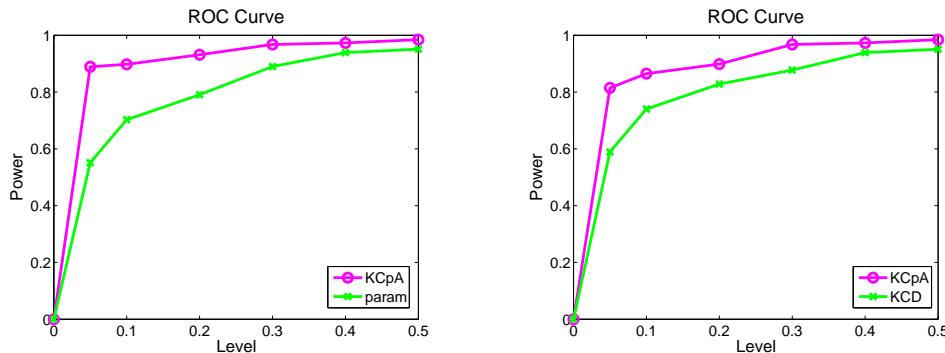

Figure 2: Comparison of ROC curves for task segmentation from BCI data (left), and pop songs segmentation (right).

**Pop song segmentation**     Indexation of music signals aims to provide a temporal segmentation into several sections with different dynamic or tonal or timbral characteristics. We investigated the performance of KCpA on a database of 100 full-length "pop music" signals, whose manual segmentation is available. In Figure 2 (right), we provide the respective ROC-curves of KCD of [23] and KCpA. Our approach is indeed competitive in this context.

## 8   Conclusion

We proposed a principled approach for the change-point analysis of a time-series of independent observations. It provides a powerful testing procedure for testing the presence of a change in distribution in a sample. Moreover, we saw in experiments that it also allows to accurately estimate the change-point when a change occurs. We are currently exploring several extensions of KCpA. Since experimental results are promising on real data, in which the assumption of independence is rather unrealistic, it is worthwhile to analyze the effect of dependence on the large-sample behaviour of our

test statistic, and explain why the test statistic remains powerful even for (weakly) dependent data. We are also investigating *adaptive* versions of the change-point analysis, in which the regularization parameter $\gamma$ and the reproducing kernel $k$ are learned from the data.

## Acknowledgments

This work has been supported by Agence Nationale de la Recherche under contract ANR-06-BLAN-0078 KERNSIG.

## Footnotes

[1]see http://ida.first.fraunhofer.de/projects/bci/competition_iii/

## References

[1] F. De la Torre Frade, J. Campoy, and J. F. Cohn. Temporal segmentation of facial behavior. In *ICCV*, 2007.

[2] J. Lafferty, A. McCallum, and F. Pereira. Conditional random fields: Probabilistic models for segmenting and labeling sequence data. In *Proc. ICML*, 2001.

[3] O. Cappé, E. Moulines, and T. Ryden. *Inference in Hidden Markov Models*. Springer, 2005.

[4] J. Chen and A.K. Gupta. *Parametric Statistical Change-point Analysis*. Birkhäuser, 2000.

[5] M. Csörgö and L. Horváth. *Limit Theorems in Change-Point Analysis*. Wiley and sons, 1998.

[6] M. Basseville and N. Nikiforov. *Detection of abrupt changes*. Prentice-Hall, 1993.

[7] T. L. Lai. Sequential analysis: some classical problems and new challenges. *Statistica Sinica*, 11, 2001.

[8] E. Lehmann and J. Romano. *Testing Statistical Hypotheses (3rd ed.)*. Springer, 2005.

[9] Z. Harchaoui, F. Bach, and E. Moulines. Testing for homogeneity with kernel Fisher discriminant analysis. In *Adv. NIPS*, 2007.

[10] G. Blanchard, O. Bousquet, and L. Zwald. Statistical properties of kernel principal component analysis. *Machine Learning*, 66, 2007.

[11] K. Fukumizu, F. Bach, and A. Gretton. Statistical convergence of kernel canonical correlation analysis. *JLMR*, 8, 2007.

[12] C. Gu. *Smoothing Spline ANOVA Models*. Springer, 2002.

[13] I. Steinwart, D. Hush, and C. Scovel. An explicit description of the rkhs of gaussian RBF kernels. *IEEE Trans. on Inform. Th.*, 2006.

[14] B. K. Sriperumbudur, A. Gretton, K. Fukumizu, G. R. G. Lanckriet, and B. Schölkopf. Injective hilbert space embeddings of probability measures. In *COLT*, 2008.

[15] B. James, K. L. James, and D. Siegmund. Tests for a change-point. *Biometrika*, 74, 1987.

[16] J. Shawe-Taylor and N. Cristianini. *Kernel Methods for Pattern Analysis*. Camb. UP, 2004.

[17] P. Billingsley. *Convergence of Probability Measures (2nd ed.)*. Wiley Interscience, 1999.

[18] P. Glasserman. *Monte Carlo Methods in Financial Engineering (1rst ed.)*. Springer, 2003.

[19] A. Gretton, K. Borgwardt, M. Rasch, B. Schoelkopf, and A.J. Smola. A kernel method for the two-sample problem. In *Adv. NIPS*, 2006.

[20] B. Schölkopf and A. J. Smola. *Learning with Kernels*. MIT Press, 2002.

[21] G. Blanchard and L. Zwald. Finite-dimensional projection for classification and statistical learning. *IEEE Transactions on Information Theory*, 54(9):4169–4182, 2008.

[22] Z. Harchaoui, F. Vallet, A. Lung-Yut-Fong, and O. Cappé. A regularized kernel-based approach to unsupervised audio segmentation. In *ICASSP*, 2009.

[23] F. Désobry, M. Davy, and C. Doncarli. An online kernel change detection algorithm. *IEEE Trans. on Signal Processing*, 53(8):2961–2974, August 2005.

[24] Z. Harchaoui and O. Cappé. Retrospective multiple change-point estimation with kernels. In *IEEE Workshop on Statistical Signal Processing (SSP)*, 2007.

